# Learning with ensembles: How over-fitting can be useful

**Peter Sollich**
Department of Physics
University of Edinburgh, U.K.
P.Sollich@ed.ac.uk

**Anders Krogh**[*]
NORDITA, Blegdamsvej 17
2100 Copenhagen, Denmark
krogh@sanger.ac.uk

## Abstract

We study the characteristics of learning with ensembles. Solving exactly the simple model of an ensemble of linear students, we find surprisingly rich behaviour. For learning in large ensembles, it is advantageous to use under-regularized students, which actually over-fit the training data. Globally optimal performance can be obtained by choosing the training set sizes of the students appropriately. For smaller ensembles, optimization of the ensemble weights can yield significant improvements in ensemble generalization performance, in particular if the individual students are subject to noise in the training process. Choosing students with a wide range of regularization parameters makes this improvement robust against changes in the unknown level of noise in the training data.

## 1   INTRODUCTION

An ensemble is a collection of a (finite) number of neural networks or other types of predictors that are trained for the same task. A combination of many different predictors can often improve predictions, and in statistics this idea has been investigated extensively, see *e.g.* [1, 2, 3]. In the neural networks community, ensembles of neural networks have been investigated by several groups, see for instance [4, 5, 6, 7]. Usually the networks in the ensemble are trained independently and then their predictions are combined.

In this paper we study an ensemble of linear networks trained on different but overlapping training sets. The limit in which all the networks are trained on the full data set and the one where all the data sets are different has been treated in [8]. In this paper we treat the case of intermediate training set sizes and overlaps

---

[*]Present address: The Sanger Centre, Hinxton, Cambs CB10 1RQ, UK.

exactly, yielding novel insights into ensemble learning. Our analysis also allows us to study the effect of regularization and of having different predictors in an ensemble.

## 2   GENERAL FEATURES OF ENSEMBLE LEARNING

We consider the task of approximating a target function $f_0$ from $R^N$ to $R$. It will be assumed that we can only obtain noisy samples of the function, and the (now stochastic) target function will be denoted $y(\mathbf{x})$. The inputs $\mathbf{x}$ are taken to be drawn from some distribution $P(\mathbf{x})$. Assume now that an ensemble of $K$ independent predictors $f_k(\mathbf{x})$ of $y(\mathbf{x})$ is available. A weighted ensemble average is denoted by a bar, like

$$\overline{f}(\mathbf{x}) = \sum_k \omega_k f_k(\mathbf{x}), \tag{1}$$

which is the final output of the ensemble. One can think of the weight $\omega_k$ as the belief in predictor $k$ and we therefore constrain the weights to be positive and sum to one. For an input $\mathbf{x}$ we define the error of the ensemble $\epsilon(\mathbf{x})$, the error of the $k$th predictor $\epsilon_k(\mathbf{x})$, and its *ambiguity* $a_k(\mathbf{x})$

$$\epsilon(\mathbf{x}) = (y(\mathbf{x}) - \overline{f}(\mathbf{x}))^2 \tag{2}$$

$$\epsilon_k(\mathbf{x}) = (y(\mathbf{x}) - f_k(\mathbf{x}))^2 \tag{3}$$

$$a_k(\mathbf{x}) = (f_k(\mathbf{x}) - \overline{f}(\mathbf{x}))^2. \tag{4}$$

The ensemble error can be written as $\epsilon(\mathbf{x}) = \overline{\epsilon}(\mathbf{x}) - \overline{a}(\mathbf{x})$ [7], where $\overline{\epsilon}(\mathbf{x}) = \sum_k \omega_k \epsilon_k(\mathbf{x})$ is the average error over the individual predictors and $\overline{a}(\mathbf{x}) = \sum_k \omega_k a_k(\mathbf{x})$ is the average of their ambiguities, which is the variance of the output over the ensemble. By averaging over the input distribution $P(\mathbf{x})$ (and implicitly over the target outputs $y(\mathbf{x})$), one obtains the ensemble *generalization error*

$$\epsilon = \overline{\epsilon} - \overline{a} \tag{5}$$

where $\epsilon(\mathbf{x})$ averaged over $P(\mathbf{x})$ is simply denoted $\epsilon$, and similarly for $\overline{\epsilon}$ and $\overline{a}$. The first term on the right is the weighted average of the generalization errors of the individual predictors, and the second is the weighted average of the ambiguities, which we refer to as the ensemble ambiguity. An important feature of equation (5) is that it separates the generalization error into a term that depends on the generalization errors of the individual students and another term that contains *all correlations* between the students. The latter can be estimated entirely from *unlabeled data, i.e.,* without any knowledge of the target function to be approximated. The relation (5) also shows that the more the predictors differ, the lower the error will be, provided the individual errors remain constant.

In this paper we assume that the predictors are trained on a sample of $p$ examples of the target function, $(\mathbf{x}^\mu, y^\mu)$, where $y^\mu = f_0(\mathbf{x}^\mu) + \eta^\mu$ and $\eta^\mu$ is some additive noise $(\mu = 1, \ldots, p)$. The predictors, to which we refer as students in this context because they learn the target function from the training examples, need not be trained on all the available data. In fact, since training on different data sets will generally increase the ambiguity, it is possible that training on subsets of the data will *improve* generalization. An additional advantage is that, by holding out for each student a different part of the total data set for the purpose of testing, one can use the whole data set for training the ensemble while still getting an unbiased estimate of the ensemble generalization error. Denoting this estimate by $\hat{\epsilon}$, one has

$$\hat{\epsilon} = \overline{\epsilon_{\text{test}}} - \hat{\overline{a}} \tag{6}$$

where $\overline{\epsilon_{\text{test}}} = \sum_k \omega_k \epsilon_{\text{test},k}$ is the average of the students' test errors. As already pointed out, the estimate $\hat{\overline{a}}$ of the ensemble ambiguity can be found from unlabeled data.

So far, we have not mentioned how to find the weights $\omega_k$. Often uniform weights are used, but optimization of the weights in some way is tempting. In [5, 6] the training set was used to perform the optimization, *i.e.*, the weights were chosen to minimize the ensemble training error. This can easily lead to over-fitting, and in [7] it was suggested to minimize the estimated generalization error (6) instead. If this is done, the estimate (6) acquires a bias; intuitively, however, we expect this effect to be small for large ensembles.

## 3  ENSEMBLES OF LINEAR STUDENTS

In preparation for our analysis of learning with ensembles of linear students we now briefly review the case of a single linear student, sometimes referred to as 'linear perceptron learning'. A linear student implements the input-output mapping

$$f(\mathbf{x}) = \frac{1}{\sqrt{N}}\mathbf{w}^{\mathrm{T}}\mathbf{x}$$

parameterized in terms of an $N$-dimensional parameter vector $\mathbf{w}$ with real components; the scaling factor $1/\sqrt{N}$ is introduced here for convenience, and $\ldots^{\mathrm{T}}$ denotes the transpose of a vector. The student parameter vector $\mathbf{w}$ should not be confused with the ensemble weights $\omega_k$. The most common method for training such a linear student (or parametric inference models in general) is minimization of the sum-of-squares training error

$$E = \sum_{\mu}(y^{\mu} - f(\mathbf{x}^{\mu}))^2 + \lambda\mathbf{w}^2$$

where $\mu = 1,\ldots,p$ numbers the training examples. To prevent the student from fitting noise in the training data, a weight decay term $\lambda\mathbf{w}^2$ has been added. The size of the weight decay parameter $\lambda$ determines how strongly large parameter vectors are penalized; large $\lambda$ corresponds to a stronger *regularization* of the student.

For a linear student, the global minimum of $E$ can easily be found. However, in practical applications using non-linear networks, this is generally not true, and training can be thought of as a stochastic process yielding a different solution each time. We crudely model this by considering white noise added to gradient descent updates of the parameter vector $\mathbf{w}$. This yields a limiting distribution of parameter vectors $P(\mathbf{w}) \propto \exp(-E/2T)$, where the 'temperature' $T$ measures the amount of noise in the training process.

We focus our analysis on the 'thermodynamic limit' $N \to \infty$ at constant normalized number of training examples, $\alpha = p/N$. In this limit, quantities such as the training or generalization error become self-averaging, *i.e.*, their averages over all training sets become identical to their typical values for a particular training set. Assume now that the training inputs $\mathbf{x}^{\mu}$ are chosen randomly and independently from a Gaussian distribution $P(\mathbf{x}) \propto \exp(-\frac{1}{2}\mathbf{x}^2)$, and that training outputs are generated by a linear target function corrupted by additive noise, *i.e.*, $y^{\mu} = \mathbf{w}_0^{\mathrm{T}}\mathbf{x}^{\mu}/\sqrt{N} + \eta^{\mu}$, where the $\eta^{\mu}$ are zero mean noise variables with variance $\sigma^2$. Fixing the length of the parameter vector of the target function to $\mathbf{w}_0^2 = N$ for simplicity, the generalization error of a linear student with weight decay $\lambda$ and learning noise $T$ becomes [9]

$$\epsilon = (\sigma^2 + T)G + \lambda(\sigma^2 - \lambda)\frac{\partial G}{\partial\lambda}. \tag{7}$$

On the r.h.s. of this equation we have dropped the term arising from the noise on the target function alone, which is simply $\sigma^2$, and we shall follow this convention throughout. The 'response function' $G$ is [10, 11]

$$G = G(\alpha, \lambda) = (1 - \alpha - \lambda + \sqrt{(1 - \alpha - \lambda)^2 + 4\lambda})/2\lambda. \tag{8}$$

For zero training noise, $T = 0$, and for any $\alpha$, the generalization error (7) is minimized when the weight decay is set to $\lambda = \sigma^2$; its value is then $\sigma^2 G(\alpha, \sigma^2)$, which is the minimum achievable generalization error [9].

## 3.1  ENSEMBLE GENERALIZATION ERROR

We now consider an ensemble of $K$ linear students with weight decays $\lambda_k$ and learning noises $T_k$ $(k = 1 \ldots K)$. Each student has an ensemble weight $\omega_k$ and is trained on $N\alpha_k$ training examples, with students $k$ and $l$ sharing $N\alpha_{kl}$ training examples (of course, $\alpha_{kk} = \alpha_k$). As above, we consider noisy training data generated by a linear target function. The resulting ensemble generalization error can be calculated by diagrammatic [10] or response function [11] methods. We refer the reader to a forthcoming publication for details and only state the result:

$$\epsilon = \sum_{kl} \omega_k \omega_l \epsilon_{kl} \tag{9}$$

where

$$\epsilon_{kl} = \frac{\rho_k \rho_l + \sigma^2 (1 - \rho_k)(1 - \rho_l)\alpha_{kl}/(\alpha_k \alpha_l)}{1 - (1 - \rho_k)(1 - \rho_l)\alpha_{kl}/(\alpha_k \alpha_l)} + \frac{T_k}{\lambda_k}\rho_k \delta_{kl}. \tag{10}$$

Here $\rho_k$ is defined as $\rho_k = \lambda_k G(\alpha_k, \lambda_k)$. The Kronecker delta in the last term of (10) arises because the training noises of different students are uncorrelated. The generalization errors and ambiguities of the individual students are

$$\epsilon_k = \epsilon_{kk} \qquad a_k = \epsilon_{kk} - 2\sum_l \omega_l \epsilon_{kl} + \sum_{lm} \omega_l \omega_m \epsilon_{lm};$$

the result for the $\epsilon_k$ can be shown to agree with the single student result (7). In the following sections, we shall explore the consequences of the general result (9). We will concentrate on the case where the training set of each student is sampled randomly from the total available data set of size $N\alpha$. For the overlap of the training sets of students $k$ and $l$ $(k \neq l)$ one then has $\alpha_{kl}/\alpha = (\alpha_k/\alpha)(\alpha_l/\alpha)$ and hence

$$\alpha_{kl} = \alpha_k \alpha_l/\alpha \tag{11}$$

up to fluctuations which vanish in the thermodynamic limit. For finite ensembles one can construct training sets for which $\alpha_{kl} < \alpha_k \alpha_l/\alpha$. This is an advantage, because it results in a smaller generalization error, but for simplicity we use (11).

## 4  LARGE ENSEMBLE LIMIT

We now use our main result (9) to analyse the generalization performance of an ensemble with a large number $K$ of students, in particular when the size of the training sets for the individual students are chosen optimally. If the ensemble weights $\omega_k$ are approximately uniform $(\omega_k \approx 1/K)$ the off-diagonal elements of the matrix $(\epsilon_{kl})$ dominate the generalization error for large $K$, and the contributions from the training noises $T_k$ are suppressed. For the special case where all students are identical and are trained on training sets of identical size, $\alpha_k = (1 - c)\alpha$, the ensemble generalization error is shown in Figure 1(left). The minimum at a nonzero value of $c$, which is the fraction of the total data set held out for testing each student, can clearly be seen. This confirms our intuition: when the students are trained on smaller, less overlapping training sets, the increase in error of the individual students can be more than offset by the corresponding increase in ambiguity.

The optimal training set sizes $\alpha_k$ can be calculated analytically:

$$c_k \equiv 1 - \alpha_k/\alpha = \frac{1 - \lambda_k/\sigma^2}{1 + G(\alpha, \sigma^2)}. \tag{12}$$

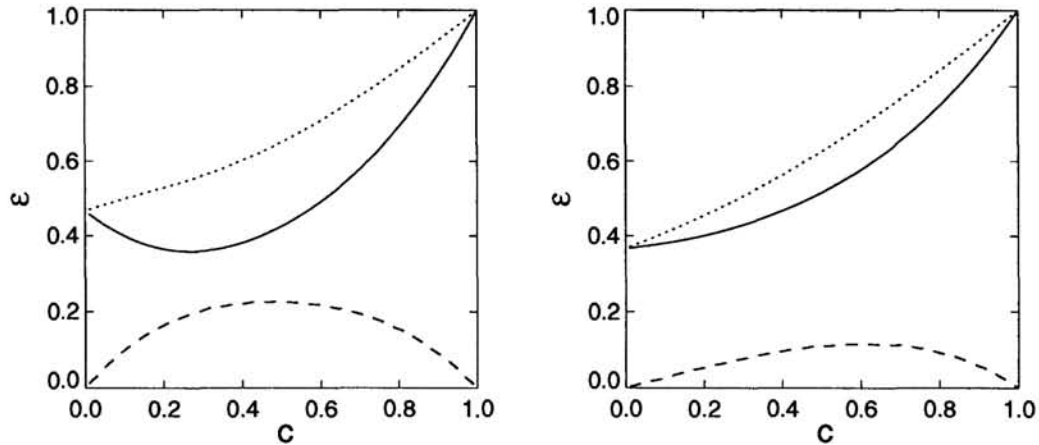

Figure 1: Generalization error and ambiguity for an infinite ensemble of identical students. Solid line: ensemble generalization error, $\epsilon$; dotted line: average generalization error of the individual students, $\bar{\epsilon}$; dashed line: ensemble ambiguity, $\bar{a}$. For both plots $\alpha = 1$ and $\sigma^2 = 0.2$. The left plot corresponds to under-regularized students with $\lambda = 0.05 < \sigma^2$. Here the generalization error of the ensemble has a minimum at a nonzero value of $c$. This minimum exists whenever $\lambda < \sigma^2$. The right plot shows the case of over-regularized students ($\lambda = 0.3 > \sigma^2$), where the generalization error is minimal at $c = 0$.

The resulting generalization error is $\epsilon = \sigma^2 G(\alpha, \sigma^2) + O(1/K)$, which is the globally minimal generalization error that can be obtained using all available training data, as explained in Section 3. Thus, *a large ensemble with optimally chosen training set sizes can achieve globally optimal generalization performance*. However, we see from (12) that a valid solution $c_k > 0$ exists only for $\lambda_k < \sigma^2$, *i.e.*, if the ensemble is under-regularized. This is exemplified, again for an ensemble of identical students, in Figure 1(right), which shows that for an over-regularized ensemble the generalization error is a monotonic function of $c$ and thus minimal at $c = 0$.

We conclude this section by discussing how the adaptation of the training set sizes could be performed in practice, for simplicity confining ourselves to an ensemble of identical students, where only one parameter $c = c_k = 1 - \alpha_k/\alpha$ has to be adapted. If the ensemble is under-regularized one expects a minimum of the generalization error for some nonzero $c$ as in Figure 1. One could, therefore, start by training all students on a large fraction of the total data set (corresponding to $c \approx 0$), and then gradually and randomly remove training examples from the students' training sets. Using (6), the generalization error of each student could be estimated by their performance on the examples on which they were not trained, and one would stop removing training examples when the estimate stops decreasing. The resulting estimate of the generalization error will be slightly biased; however, for a large enough ensemble the risk of a strongly biased estimate from systematically testing all students on too 'easy' training examples seems small, due to the random selection of examples.

## 5   REALISTIC ENSEMBLE SIZES

We now discuss some effects that occur in learning with ensembles of 'realistic' sizes. In an over-regularized ensemble nothing can be gained by making the students more diverse by training them on smaller, less overlapping training sets. One would also

Figure 2: The generalization error of an ensemble with 10 identical students as a function of the test set fraction $c$. From bottom to top the curves correspond to training noise $T = 0, 0.1, 0.2, \ldots, 1.0$. The star on each curve shows the error of the optimal single perceptron (*i.e.*, with optimal weight decay for the given $T$) trained on all examples, which is independent of $c$. The parameters for this example are: $\alpha = 1$, $\lambda = 0.05$, $\sigma^2 = 0.2$.

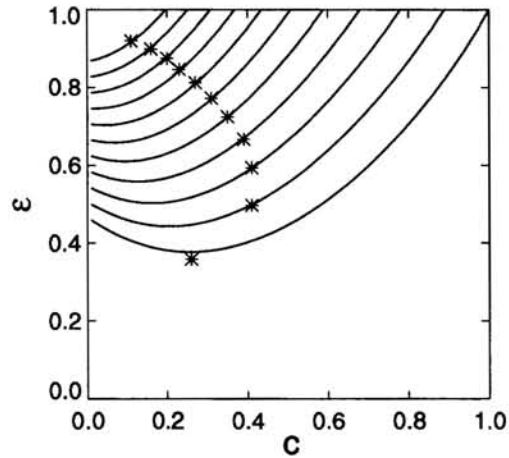

expect this kind of 'diversification' to be unnecessary or even counterproductive when the training noise is high enough to provide sufficient 'inherent' diversity of students. In the large ensemble limit, we saw that this effect is suppressed, but it does indeed occur in finite ensembles. Figure 2 shows the dependence of the generalization error on $c$ for an ensemble of 10 identical, under-regularized students with identical training noises $T_k = T$. For small $T$, the minimum of $\epsilon$ at nonzero $c$ persists. For larger $T$, $\epsilon$ is monotonically increasing with $c$, implying that further diversification of students beyond that caused by the learning noise is wasteful. The plot also shows the performance of the optimal single student (with $\lambda$ chosen to minimize the generalization error at the given $T$), demonstrating that the ensemble can perform significantly better by effectively averaging out learning noise.

For realistic ensemble sizes the presence of learning noise generally reduces the potential for performance improvement by choosing optimal *training set sizes*. In such cases one can still adapt the *ensemble weights* to optimize performance, again on the basis of the estimate of the ensemble generalization error (6). An example is

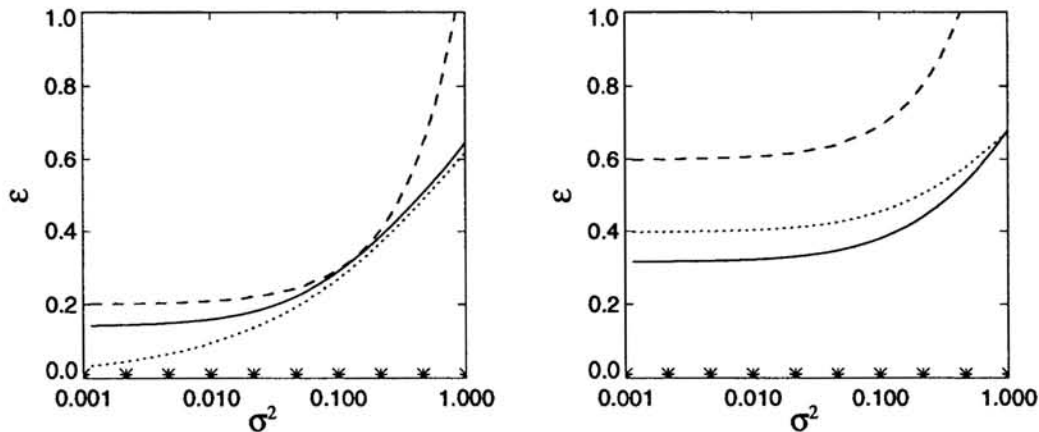

Figure 3: The generalization error of an ensemble of 10 students with different weight decays (marked by stars on the $\sigma^2$-axis) as a function of the noise level $\sigma^2$. Left: training noise $T = 0$; right: $T = 0.1$. The dashed lines are for the ensemble with uniform weights, and the solid line is for optimized ensemble weights. The dotted lines are for the optimal single perceptron trained on all data. The parameters for this example are: $\alpha = 1$, $c = 0.2$.

shown in Figure 3 for an ensemble of size $K = 10$ with the weight decays $\lambda_k$ equally spaced on a logarithmic axis between $10^{-3}$ and 1. For both of the temperatures $T$ shown, the ensemble with uniform weights performs worse than the optimal single student. With weight optimization, the generalization performance approaches that of the optimal single student for $T = 0$, and is actually better at $T = 0.1$ over the whole range of noise levels $\sigma^2$ shown. Even the best single student from the ensemble can never perform better than the optimal single student, so combining the student outputs in a weighted ensemble average is superior to simply choosing the best member of the ensemble by cross-validation, *i.e.*, on the basis of its estimated generalization error. The reason is that the ensemble average suppresses the learning noise on the individual students.

## 6    CONCLUSIONS

We have studied ensemble learning in the simple, analytically solvable scenario of an ensemble of linear students. Our main findings are: In large ensembles, one should use under-regularized students in order to maximize the benefits of the variance-reducing effects of ensemble learning. In this way, the globally optimal generalization error on the basis of *all* the available data can be reached by optimizing the training set sizes of the individual students. At the same time an estimate of the generalization error can be obtained. For ensembles of more realistic size, we found that for students subjected to a large amount of noise in the training process it is unnecessary to increase the diversity of students by training them on smaller, less overlapping training sets. In this case, optimizing the ensemble weights can still yield substantially better generalization performance than an optimally chosen single student trained on all data with the same amount of training noise. This improvement is most insensitive to changes in the unknown noise levels $\sigma^2$ if the weight decays of the individual students cover a wide range. We expect most of these conclusions to carry over, at least qualitatively, to ensemble learning with nonlinear models, and this correlates well with experimental results presented in [7].

## References

[1] C. Granger, Journal of Forecasting **8**, 231 (1989).
[2] D. Wolpert, Neural Networks **5**, 241 (1992).
[3] L. Breimann, Tutorial at *NIPS 7* and personal communication.
[4] L. Hansen and P. Salamon, IEEE Trans. Pattern Anal. and Mach. Intell. **12**, 993 (1990).
[5] M. P. Perrone and L. N. Cooper, in *Neural Networks for Speech and Image processing*, ed. R. J. Mammone (Chapman-Hall, 1993).
[6] S. Hashem: Optimal Linear Combinations of Neural Networks. Tech. Rep. PNL-SA-25166, submitted to Neural Networks (1995).
[7] A. Krogh and J. Vedelsby, in *NIPS 7*, ed. G. Tesauro *et al.*, p. 231 (MIT Press, 1995).
[8] R. Meir, in *NIPS 7*, ed. G. Tesauro *et al.*, p. 295 (MIT Press, 1995).
[9] A. Krogh and J. A. Hertz, J. Phys. A **25**, 1135 (1992).
[10] J. A. Hertz, A. Krogh, and G. I. Thorbergsson, J. Phys. A **22**, 2133 (1989).
[11] P. Sollich, J. Phys. A **27**, 7771 (1994).